# Color Opponency Constitutes A Sparse Representation For the Chromatic Structure of Natural Scenes

Te-Won Lee,* Thomas Wachtler and Terrence Sejnowski

Institute for Neural Computation, University of California, San Diego &
Computational Neurobiology Laboratory, The Salk Institute
10010 N. Torrey Pines Road
La Jolla, California 92037, USA
{tewon,thomas,terry}@salk.edu

## Abstract

The human visual system encodes the chromatic signals conveyed by the three types of retinal cone photoreceptors in an opponent fashion. This color opponency has been shown to constitute an efficient encoding by spectral decorrelation of the receptor signals. We analyze the spatial and chromatic structure of natural scenes by decomposing the spectral images into a set of linear basis functions such that they constitute a representation with minimal redundancy. Independent component analysis finds the basis functions that transforms the spatiochromatic data such that the outputs (activations) are statistically as independent as possible, i.e. least redundant. The resulting basis functions show strong opponency along an achromatic direction (luminance edges), along a blue-yellow direction, and along a red-blue direction. Furthermore, the resulting activations have very sparse distributions, suggesting that the use of color opponency in the human visual system achieves a highly efficient representation of colors. Our findings suggest that color opponency is a result of the properties of natural spectra and not solely a consequence of the overlapping cone spectral sensitivities.

## 1 Statistical structure of natural scenes

Efficient encoding of visual sensory information is an important task for information processing systems and its study may provide insights into coding principles of biological visual systems. An important goal of sensory information processing

Electronic version available at www.cnl.salk.edu/~tewon.

is to transform the input signals such that the redundancy between the inputs is reduced. In natural scenes, the image intensity is highly predictable from neighboring measurements and an efficient representation preserves the information while the neuronal output is minimized. Recently, several methods have been proposed for finding efficient codes for achromatic images of natural scenes [1, 2, 3, 4]. While luminance dominates the structure of the visual world, color vision provides important additional information about our environment. Therefore, we are interested in efficient, i.e. redundancy reducing representations for the chromatic structure of natural scenes.

## 2 Learning efficient representation for chromatic image

Our goal was to find efficient representations of the chromatic sensory information such that its spatial and chromatic redundancy is reduced significantly. The method we used for finding statistically efficient representations is independent component analysis (ICA). ICA is a way of finding a linear non-orthogonal co-ordinate system in multivariate data that minimizes mutual information among the axial projections of the data. The directions of the axes of this co-ordinate system (basis functions) are determined by both second and higher-order statistics of the original data, compared to Principal Component Analysis (PCA) which is used solely in second order statistics and has orthogonal basis functions. The goal of ICA is to perform a linear transform which makes the resulting source outputs as statistically independent from each other as possible [5]. ICA assumes an unknown source vector $\mathbf{s}$ with mutually independent components $s_i$. A small patch of the observed image is stretched into a vector $\mathbf{x}$ that can be represented as a linear combination of sources components $s_i$ such that

$$\mathbf{x} = \mathbf{As},\qquad(1)$$

where $\mathbf{A}$ is a scalar square matrix and the columns of $\mathbf{A}$ are the basis functions. Since $\mathbf{A}$ and $\mathbf{s}$ are unknown the goal of ICA is to adapt the basis functions by estimating $\mathbf{s}$ so that the individual components $s_i$ are statistically independent and this adaptation process minimizes the mutual information between the components $s_i$. A learning algorithm can be derived using the information maximization principle [5] or the maximum likelihood estimation (MLE) method which can be shown to be equivalent in this case. In our experiments, we used the infomax learning rule with natural gradient extension and the learning algorithm for the basis functions is

$$\Delta\mathbf{A} \propto \mathbf{A}\left[\mathbf{I} - \varphi(\mathbf{s})\mathbf{s}^T\right].\qquad(2)$$

where $\mathbf{I}$ is the identity matrix, $\varphi(\mathbf{s}) = -\frac{\partial p(\mathbf{s})/\partial \mathbf{s}}{p(\mathbf{s})}$ and $\mathbf{s}^T$ denotes the matrix transpose of $\mathbf{s}$. $\Delta\mathbf{A}$ is the change of the basis functions that is added to $\mathbf{A}$. The change in $\Delta\mathbf{A}$ will converge to zero once the adaptation process is complete. Note that $\varphi(\mathbf{s})$ requires a density model for $p(s_i)$. We used a parametric exponential power density $p(s_i) \propto \exp(-|s_i|^{q_i})$ and simultaneously updated its shape by inferring the value $q_i$ to match the distribution of the estimated sources [6]. This is accomplished by finding the maximum posteriori value of $q_i$ given the observed data. The ICA algorithm can thus characterize a wide class of statistical distributions including uniform, Gaussian, Laplacian, and other so-called sub- and super-Gaussian densities. In other words, our experiments do not constrain the coefficients to have a

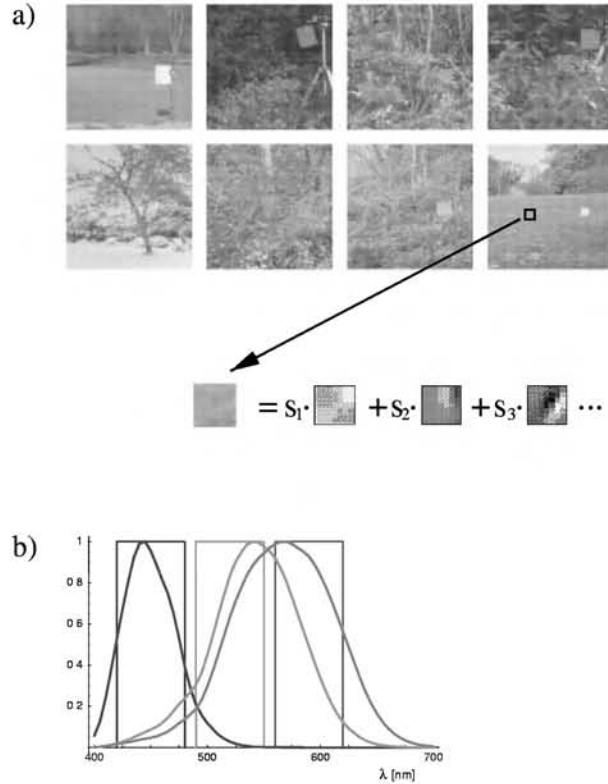

Figure 1: Linear decomposition of an observed spectral image patch into its basis functions.

sparse distribution, unlike some previous methods [1, 2]. The algorithm converged to a solution of maximal independence and the distributions of the coefficients were approximated by exponential power densities.

We investigated samples of spectral images of natural scenes as illustrated in Figure 1. We analyzed a set of hyperspectral images [7] with a size of 256×256 pixels. Each pixel is represented by radiance values for 31 wavebands of 10 nm width, sampled in 10 nm steps between 400 and 700 nm. The pixel size corresponds to 0.056×0.056 deg of visual angle. The images were recorded around Bristol, either outdoors, or inside the glass houses of Bristol Botanical Gardens. We chose eight of these images which had been obtained outdoors under apparently different illumination conditions. The vector of 31 spectral radiance values of each pixel was converted to a vector of 3 cone excitation values whose components were the inner products of the radiance vector with the vectors of L-, M-, and S-cone sensitivity values [8], respectively. From the entire image data set, 7×7 pixel image patches were chosen randomly, yielding $7 \times 7 \times 3 = 147$ dimensional vectors. The learning process was done in 500 steps, each using a set of spectra of 40000 image patches, 5000 chosen randomly from each of the eight images. A set of basis functions for 7x7 pixel patches was obtained, with each pixel containing the logarithms of the excitations of the three human cone photoreceptors that represented the receptor signals in the human retina [8, 9]. To visualize the learned basis functions, we

used the method by Ruderman et al.[9] and plotted for each basis function a $7\times7$ pixel matrix, with the color of each pixel indicating the combination of L, M, and S cone responses as follows. The values for each patch were normalized to values between 0 and 255, with 0 cone excitation corresponding to a value of 128. Thus, the R, G, and B components of each pixel represent the relative excitations of L, M, and S cones, respectively. To further illustrate the chromatic properties of the basis functions, we convert the L, M, S vector of each pixel to its projection onto the isoluminant plane of a cone-opponent color space similar to the color spaces of MacLeod and Boynton[10] and Derrington et al[11]. In our plots, the horizontal axis corresponds to the response of an L cone versus M cone opponent mechanism, the vertical axis corresponds to S cone modulation. For each pixel of the basis functions, a point is plotted at its corresponding location in that color space. The color of the points are the same as used for the pixels in the top part of the figure. Thus, although only the projection onto the isoluminant plane is shown, the third dimension (i.e., luminance) can be inferred by the brightness of the points. Figure 2a shows the learned ICA basis functions in a pseudo color representation. Figure 2b shows the color space coordinates of the chromaticities of the pixels in each basis function. The PCA basis functions and their corresponding color space coordinates are shown in Figure 2c and 2d respectively. Both representations are in order of decreasing $L_2$-norm. The PCA results show a global spatial representation and their opponent basis functions lie mostly along the coordinate axes of the cone-opponent color space. In addition, there are functions that imply mixtures of non-opponent colors. In contrast to PCA basis functions, the ICA basis functions are localized and oriented. When ordered by decreasing $L_2$-norm, achromatic basis functions tend to appear before chromatic basis functions. This reflects the fact that in the natural environment, luminance variations are generally larger than chromatic variations [7]. The achromatic basis functions are localized and oriented, similar to those found in the analysis of grayscale natural images [1, 2]. Most of the chromatic basis functions, particularly those with strong contributions, are color opponent, i.e., the chromaticities of their pixels lie roughly along a line through the origin of our color space. Most chromatic basis functions with relatively high contributions are modulated between light blue and dark yellow, in the plane defined by luminance and S-cone modulation. Those with lower $L_2$-norm are highly localized, but still are mostly oriented. There are other chromatic basis functions with tilted orientations, corresponding to blue versus orange colors. The chromaticities of these basis functions occupy mainly the second and fourth quadrant. The basis functions with lowest contributions are less strictly aligned in color space, but still tend to be color opponent, mostly along a bluish-green/orange direction. There are no basis functions with chromaticities along the horizontal axis, corresponding to pure L versus M cone opponency, like PCA basis functions in Figure 2d [9]. The tilted orientations of the opponency axes most likely reflects the distribution of the chromaticities in our images. In natural images, L-M and S coordinates in our color space are negatively correlated [12]. ICA finds the directions that correspond to maximally decorrelated signals, i.e. extracts statistical structure of the inputs. PCA did not yield basis functions in these directions, probably because it is limited by the orthogonality constraint. While it is known that chromatic properties of neurons in the lateral geniculate nucleus (LGN) of primates correspond to variations along the axes of cone-opponency ('cardinal axes') [11], cortical neurons show sensitivities for intermediate directions [13]. Since the results of PCA and ICA,

respectively, match these differences qualitatively, we suspect that opponent coding along the 'cardinal directions' of cone opponency is used by the visual system to transmit reliably visual information to the cortex, where the information is recoded in order to better reflect the statistical structure of the environment [14].

## 3  Discussion

This result shows that the independence criterion alone is sufficient to learn efficient image codes. Although no sparseness constraint was used, the obtained coefficients are extremely sparse, i.e. the data $\mathbf{x}$ are encoded in the sources $\mathbf{s}$ in such a way that the coefficients of $\mathbf{s}$ are mostly around zero; there is only a small percentage of informative values (non-zero coefficients). From an information coding perspective this assumes that we can encode and decode the chromatic image patches with only a small percentage of the basis functions. In contrast, Gaussian densities are not sparsely distributed and a large portion of the basis functions is required to represent the chromatic images. The normalized kurtosis value is one measure of sparseness and the average kurtosis value was 19.7 for ICA, and 6.6 for PCA. Interestingly the basis functions in Figure 2a produced only sparse coefficients except for basis function 7 (green basis function) that resulted in a nearly uniform distribution, suggesting that this basis function is active almost all the time. The reason may be that a green color component is present in almost all image patches of the natural scenes. We repeated the experiment with different ICA methods and obtained similar results. The basis functions obtained with the exponential power distributions or the simple Laplacian prior were statistically most efficient. In this sense, the basis functions that produce sparse distributions are statistically efficient codes. To quantitatively measure the encoding difference we compared the coding efficiency between ICA and PCA using Shannon's theorem to obtain a lower bound on the number of bits required to encode a spatiochromatic pattern [4]. The average number of bits required to encode 40000 patches randomly selected from the 8 images in Figure 1 with a fixed noise coding precision of $\sigma_x = 0.059$ was 1.73 bits for ICA and 4.46 bits for PCA. Note that the encoding difference for achromatic image patches using ICA and PCA is about 20% in favor of ICA [4]. The encoding difference in the chromatic case is significantly higher ($> 100\%$) and suggests that there is a large amount of chromatic redundancy in the natural scenes. To verify our findings, we computed the average pairwise mutual information $I$ in the original data ($I_x = 0.1522$), the PCA representation ($I_{PCA} = 0.0123$) and the ICA representation ($I_{ICA} = 0.0093$). ICA was able to further reduce the redundancy between its components, and its basis functions therefore represent more efficient codes.

In general, the ICA results support the argument that basis functions for efficient coding of chromatic natural images are non-orthogonal. In order to determine whether the color opponency is merely a result of correlation in the receptor signals due to the strong overlap of the photoreceptor sensitivities [15], we repeated the analysis, this time assuming hypothetical receptor sensitivities which do not overlap, but sample roughly in the same regions as the L-, M-, and S- cones. We used rectangular sensitivities with absorptions between 420 and 480 nm ("S"), 490 and 550 nm ("M"), and 560 and 620 nm ("L"), respectively. The resulting basis functions were as strongly color opponent as for the case of overlapping cone sensitivities. This suggests that the correlations of radiance values in natural spectra are

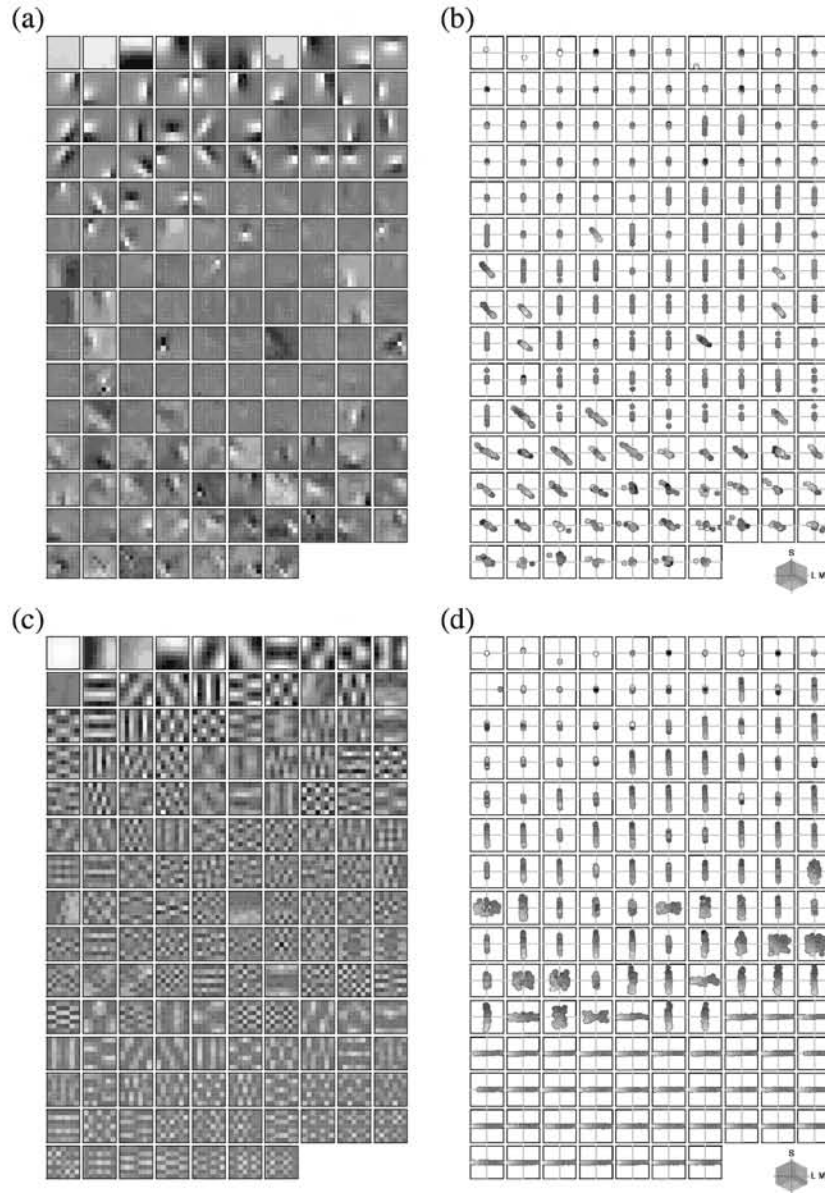

Figure 2: **(a)** 147 total ICA spatiochromatic structure of basis functions (7 by 7 pixels and 3 colors) are shown in order of decreasing $L_2$-norm, from top to bottom and left to right. The R, G, and B values of the color of each pixel correspond to the relative excitation of L-, M-, and S-cones, respectively. **(b)** Chromaticities of the ICA basis functions, plotted in cone-opponent color space coordinates. Each dot represents the coordinate of a pixel of the respective basis function, projected onto the isoluminant plane. Luminance can be inferred from the brightness of the dot. Horizontal axes: L- versus M-cone variation. Vertical axes: S-cone variation. **(c)** 147 PCA spatiochromatic basis functions and **(d)** Corresponding PCA chromaticities.

sufficiently high to require a color opponent code in order to represent the chromatic structure efficiently. In summary, our findings strongly suggest color opponency is not a mere consequence of the overlapping cone spectral sensitivities but moreover an attempt to represent the intrinsic spatiochromatic structure of natural scenes in a statistically efficient manner.

# References

[1] B. Olshausen and D. Field. Emergence of simple-cell receptive field properties by learning a sparse code for natural images. *Nature*, 381:607–609, 1996.

[2] A. J. Bell and T. J. Sejnowski. The 'independent components' of natural scenes are edge filters. *Vision Research*, 37(23):3327–3338, 1997.

[3] J. H. van Hateren and A. van der Schaaf. Independent component filters of natural images compared with simple cells in primary visual cortex. *Proc.R.Soc.Lond. B*, 265:359–366, 1998.

[4] M.S. Lewicki and B. Olshausen. A probablistic framwork for the adaptation and comparison of image codes. *J. Opt.Soc., A: Optics, Image Science and Vision*, in press, 1999.

[5] A. J. Bell and T. J. Sejnowski. An Information-Maximization Approach to Blind Separation and Blind Deconvolution. *Neural Computation*, 7:1129–1159, 1995.

[6] M.S. Lewicki. A flexible prior for independent component analysis. *Neural Computation*, submitted, 2000.

[7] C. A. Párraga, G. Brelstaff, and T. Troscianko. Color and luminance information in natural scenes. *Journal of the Optical Society of America A*, 15:563–569, 1998. (http://www.crs4.it/~gjb/ftpJOSA.html).

[8] A. Stockman, D. I. A. MacLeod, and N. E. Johnson. Spectral sensitivities of the human cones. *Journal of the Optical Society of America A*, 10:2491–2521, 1993. (http://www-cvrl.ucsd.edu).

[9] D. L. Ruderman, T. W. Cronin, and C.-C. Chiao. Statistics of cone responses to natural images: Implications for visual coding. *Journal of the Optical Society of America A*, 15:2036–2045, 1998.

[10] D. I. A. MacLeod and R. M. Boynton. Chromaticity diagram showing cone excitation by stimuli of equal luminance. *Journal of the Optical Society of America*, 69:1183–1186, 1979.

[11] A. M. Derrington, J. Krauskopf, and P. Lennie. Chromatic mechanisms in lateral geniculate nucleus of macaque. *Journal of Physiology*, 357:241–265, 1984.

[12] D. I. A. MacLeod and T. von der Twer. The pleistochrome: Optimal opponent codes for natural colors. Preprint, 1998.

[13] P. Lennie, J. Krauskopf, and G. Sclar. Chromatic mechanisms in striate cortex of macaque. *Journal of Neuroscience*, 10:649–669, 1990.

[14] D. J. Field. What is the goal of sensory coding? *Neural Computation*, 6:559–601, 1994.

[15] G. Buchsbaum and A. Gottschalk. Trichromacy, opponent colours coding and optimum colour information transmission in the retina. *Proceedings of the Royal Society London B*, 220:89–113, 1983.
